# Learning Visual Attributes

**Vittorio Ferrari** *
University of Oxford (UK)

**Andrew Zisserman**
University of Oxford (UK)

## Abstract

We present a probabilistic generative model of visual attributes, together with an efficient learning algorithm. Attributes are visual qualities of objects, such as 'red', 'striped', or 'spotted'. The model sees attributes as patterns of image segments, repeatedly sharing some characteristic properties. These can be any combination of appearance, shape, or the layout of segments within the pattern. Moreover, attributes with general appearance are taken into account, such as the pattern of alternation of *any* two colors which is characteristic for stripes. To enable learning from unsegmented training images, the model is learnt discriminatively, by optimizing a likelihood ratio.

As demonstrated in the experimental evaluation, our model can learn in a weakly supervised setting and encompasses a broad range of attributes. We show that attributes can be learnt starting from a text query to Google image search, and can then be used to recognize the attribute and determine its spatial extent in novel real-world images.

## 1 Introduction

In recent years, the recognition of object categories has become a major focus of computer vision and has shown substantial progress, partly thanks to the adoption of techniques from machine learning and the development of better probabilistic representations [1, 3]. The goal has been to recognize object categories, such as a 'car', 'cow' or 'shirt'. However, an object also has many other qualities apart from its category. A car can be *red*, a shirt *striped*, a ball *round*, and a building *tall*. These visual *attributes* are important for understanding object appearance and for describing objects to other people. Figure 1 shows examples of such attributes. Automatic learning and recognition of attributes can complement category-level recognition and therefore improve the degree to which machines perceive visual objects. Attributes also open the door to appealing applications, such as more specific queries in image search engines (e.g. a spotted skirt, rather than just any skirt). Moreover, as different object categories often have attributes in common, modeling them explicitly allows part of the learning task to be shared amongst categories, or allows previously learnt knowledge about an attribute to be transferred to a novel category. This may reduce the total number of training images needed and improve robustness. For example, learning the variability of zebra stripes under non-rigid deformations tells us a lot about the corresponding variability in striped shirts.

In this paper we propose a probabilistic generative model of visual attributes, and a procedure for learning its parameters from real-world images. When presented with a novel image, our method infers whether it contains the learnt attribute and determines the region it covers. The proposed model encompasses a broad range of attributes, from simple colors such as 'red' or 'green' to complex patterns such as 'striped' or 'checked'. Both the appearance and the shape of pattern elements (e.g. a single stripe) are explicitly modeled, along with their layout within the overall pattern (e.g. adjacent stripes are parallel). This enables our model to cover attributes defined by appearance ('red'), by shape ('round'), or by both (the black-and-white stripes of zebras). Furthermore, the model takes into account attributes with general appearance, such as stripes which are characterized by a pattern of alternation ABAB of any two colors A and B, rather than by a specific combination of colors.

Since appearance, shape, and layout are modeled explictly, the learning algorithm gains an understanding of the nature of the attribute. As another attractive feature, our method can learn in a weakly supervised setting, given images labeled only by the presence or absence of the attribute,

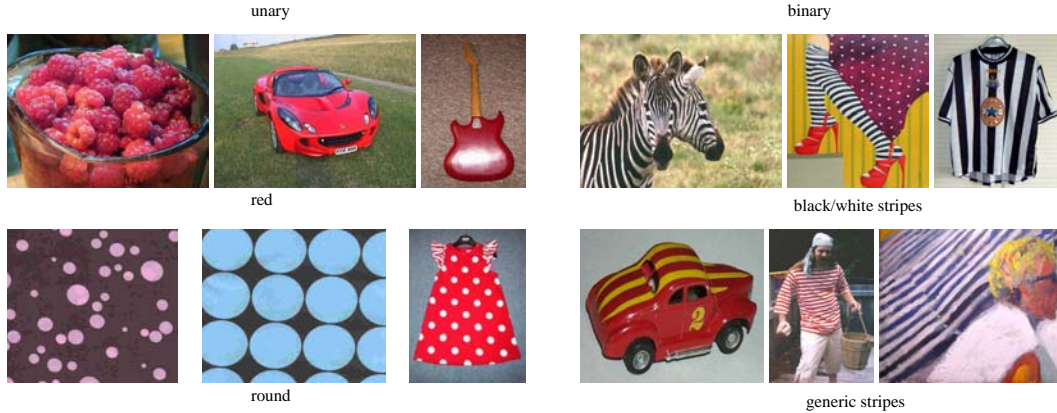

Figure 1: *Examples of different kinds of attributes. On the left we show two simple attributes, whose characteristic properties are captured by individual image segments (appearance for red, shape for round). On the right we show more complex attributes, whose basic element is a pair of segments.*

without indication of the image region it covers. The presence/absence labels can be noisy, as the training method can tolerate a considerable number of mislabeled images. This enables attributes to be learnt directly from a text specification by collecting training images using a web image search engine, such as Google-images, and querying on the name of the attribute.

Our approach is inspired by the ideas of Jojic and Caspi [4], where patterns have constant appearance within an image, but are free to change to another appearance in other images. We also follow the generative approach to learning a model from a set of images used by many authors, for example LOCUS [10]. Our parameter learning is discriminative – the benefits of this have been shown before, for example for training the constellation model of [3]. In term of functionality, the closest works to ours are those on the analysis of regular textures [5, 6]. However, they work with textures covering the entire image and focus on finding distinctive appearance descriptors. In constrast, here textures are attributes of objects, and therefore appear in complex images containing many other elements. Very few previous works appeared in this setting [7, 11]. The approach of [7] focuses on colors only, while in [11] attributes are limited to individual regions. Our method encompasses also patterns defined by pairs of regions, allowing to capture more complex attributes. Moreover, we take up the additional challenge of learning the pattern geometry.

Before describing the generative model in section 3, in the next section we briefly introduce image segments, the elementary units of measurements observed in the model.

## 2  Image segments – basic visual representation

The basic units in our attribute model are image segments extracted using the algorithm of [2]. Each segment has a uniform appearance, which can be either a color or a simple texture (e.g. sand, grain). Figure 2a shows a few segments from a typical image.

Inspired by the success of simple patches as a basis for appearance descriptors [8, 9], we randomly sample a large number of $5 \times 5$ pixel patches from all training images and cluster them using k-means [8]. The resulting cluster centers form a codebook of *patch types*. Every pixel is soft-assigned to the patch types. A segment is then represented as a normalized histogram over the patch types of the pixels it contains. By clustering the segment histograms from the training images we obtain a codebook $\mathcal{A}$ of *appearances* (figure 2b). Each entry in the codebook is a prototype segment descriptor, representing the appearance of a subset of the segments from the training set.

Each segment $s$ is then assigned the appearance $a \in \mathcal{A}$ with the smallest Bhattacharya distance to the histogram of $s$. In addition to appearance, various geometric properties of a segment are measured, summarizing its shape. In our current implementation, these are: curvedness, compactness, elongation (figure 2c), fractal dimension and area relative to the image. We also compute two properties of pairs of segments: relative orientation and relative area (figure 2d).

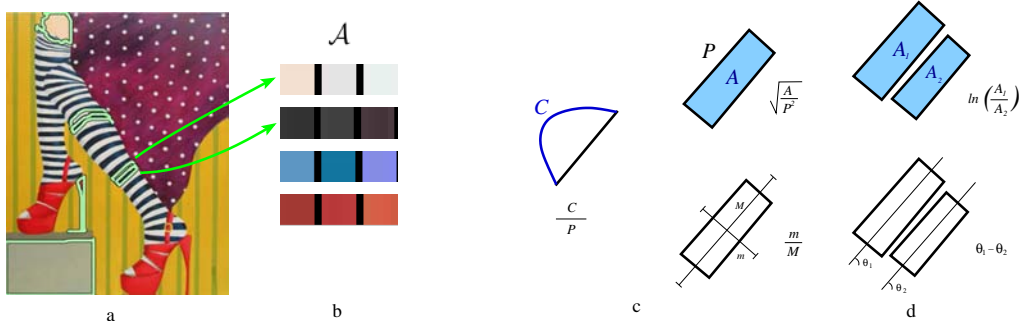

Figure 2: *Image segments as visual features. a) An image with a few segments overlaid, including two pairs of adjacent segments on a striped region. b) Each row is an entry from the appearance codebook $\mathcal{A}$ (i.e. one appearance; only 4 out of 32 are shown). The three most frequent patch types for each appearance are displayed. Two segments from the stripes are assigned to the white and black appearance respectively (arrows). c) Geometric properties of a segment: curvedness, which is the ratio between the number of contour points $C$ with curvature above a threshold and the total perimeter $P$; compactness; and elongation, which is the ratio between the minor and major moments of inertia. d) Relative geometric properties of a pair of segments: relative area and relative orientation. Notice how these measures are not symmetric (e.g. relative area is the area of the first segment wrt to the second).*

## 3 Generative models for visual attributes

Figure 1 shows various kinds of attributes. Simple attributes are entirely characterized by properties of a single segment (*unary attributes*). Some unary attributes are defined by their appearance, such as colors (e.g. red, green) and basic textures (e.g. sand, grainy). Other unary attributes are defined by a segment shape (e.g. round). All red segments have similar appearance, regardless of shape, while all round segments have similar shape, regardless of appearance. More complex attributes have a basic element composed of *two* segments (*binary attributes*). One example is the black/white stripes of a zebra, which are composed of pairs of segments sharing similar appearance *and* shape across *all* images. Moreover, the layout of the two segments is characteristic as well: they are adjacent, nearly parallel, and have comparable area. Going yet further, a *general* stripe pattern can have *any* appearance (e.g. blue/white stripes, red/yellow stripes). However, the pairs of segments forming a stripe pattern in one particular image must have the same appearance. Hence, a characteristic of general stripes is a pattern of alternation ABABAB. In this case, appearance is common within an image, but not across images.

The attribute models we present in this section encompass all aspects discussed above. Essentially, attributes are found as patterns of repeated segments, or pairs of segments, sharing some properties (geometric and/or appearance and/or layout).

### 3.1 Image likelihood.

We start by describing how the model $\mathcal{M}$ explains a whole image $I$. An image $I$ is represented by a set of segments $\{s\}$. A latent variable $f$ is associated with each segment, taking the value $f = 1$ for a foreground segment, and $f = 0$ for a background segment. Foreground segments are those on the image area covered by the attribute. We collect $f$ for all segments of $I$ into the vector $\mathbf{F}$. An image has a foreground appearance $a$, shared by all the foreground segments it contains. The likelihood of an image is

$$p(I|\mathcal{M}; \mathbf{F}, a) = \prod_{x \in I} p(x|\mathcal{M}; \mathbf{F}, a) \qquad (1)$$

where $x$ is a pixel, and $\mathcal{M}$ are the model parameters. These include $\alpha \subset \mathcal{A}$, the set of appearances allowed by the model, from which $a$ is taken. The other parameters are used to explain segments and are dicussed below. The probability of pixels is uniform within a segment, and independent across segments:

$$p(x|\mathcal{M}; \mathbf{F}, a) = p(s^x|\mathcal{M}; f, a) \qquad (2)$$

with $s^x$ the segment containing $x$. Hence, the image likelihood can be expressed as a product over the probability of each segment $s$, counted by its area $N_s$ (i.e. the number of pixels it contains)

$$p(I|\mathcal{M}; \mathbf{F}, a) = \prod_{x \in I} p(s^x|\mathcal{M}; f, a) = \prod_{s \in I} p(s|\mathcal{M}; f, a)^{N_s} \qquad (3)$$

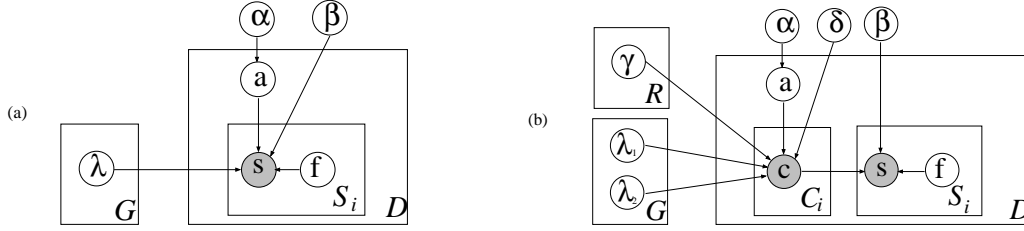

Figure 3: *a) Graphical model for unary attributes. $D$ is the number of images in the dataset, $S_i$ is the number of segments in image i, and $G$ is the total number of geometric properties considered (both active and inactive). b) Graphical model for binary attributes. $c$ is a pair of segments. $\Phi_{1,2}$ are the geometric distributions for each segment a pair. $\Psi$ are relative geometric distributions (i.e. measure properties between two segments in a pair, such as relative orientation), and there are $R$ of them in total (active and inactive). $\delta$ is the adjacency model parameter. It tells whether only adjacent pairs of segments are considered (so $p(c|\delta = 1)$ is one only iff c is a pair of adjacent segments).*

Note that $\mathbf{F}$ and $a$ are latent variables associated with a particular image, so there is a different $\mathbf{F}$ and $a$ for each image. In contrast, a single model $\mathcal{M}$ is used to explain all images.

## 3.2 Unary attributes

Segments are the only observed variables in the unary model. A segment $s = (s_a, \{s_g^j\})$ is defined by its appearance $s_a$ and shape, captured by a set of geometric measurements $\{s_g^j\}$, such as elongation and curvedness. The graphical model in figure 3a illustrates the conditional probability of image segments

$$p(s|\mathcal{M}; f, a) = \begin{cases} p(s_a|a) \cdot \prod_j p(s_g^j|\Phi^j)^{v^j} & \text{if } f = 1 \\ \beta & \text{if } f = 0 \end{cases} \tag{4}$$

The likelihood for a segment depends on the model parameters $\mathcal{M} = (\alpha, \beta, \{\lambda^j\})$, which specify a visual attribute. For each geometric property $\lambda^j = (\Phi^j, v^j)$, the model defines its distribution $\Phi^j$ over the foreground segments and whether the property is *active* or not ($v^j = 1$ or 0). Active properties are relevant for the attribute (e.g. elongation is relevant for stripes, while orientation is not) and contribute substantially to its likelihood in (4). Inactive properties instead have no impact on the likelihood (exponentiation by 0). It is the task of the learning stage to determine which properties are active and their foreground distribution.

The factor $p(s_a|a) = [s_a = a]$ is 1 for segments having the foreground appearance $a$ for this image, and 0 otherwise (thus it acts as a selector). The scalar value $\beta$ represents a simple background model: all segments assigned to the background have likelihood $\beta$. During inference and learning we want to maximize the likelihood of an image given the model over $\mathbf{F}$, which is achieved by setting $f$ to foreground when the $f = 1$ case of equation (4) is greater than $\beta$.

As an example, we give the ideal model parameters for the attribute 'red'. $\alpha$ contains the red appearance only. $\beta$ is some low value, corresponding to how likely it is for non-red segments to be assigned the red appearance. No geometric property $\{\lambda^j\}$ is active (i.e. all $v^j = 0$).

## 3.3 Binary attributes

The basic element of binary attributes is a *pair* of segments. In this section we extend the unary model to describe pairs of segments. In addition to duplicating the unary appearance and geometric properties, the extended model includes pairwise properties which do not apply to individual segments. In the graphical model of figure 3b, these are relative geometric properties $\gamma$ (area, orientation) and adjacency $\delta$, and together specify the *layout* of the attribute. For example, the orientation of a segment with respect to the other can capture the parallelism of subsequent stripe segments. Adjacency expresses whether the two segments in the pair are adjacent (like in stripes) or not (like the maple leaf and the stripes in the canadian flag). We consider two segments adjacent if they share part of the boundary. A pattern characterized by adjacent segments is more distinctive, as it is less likely to occur accidentally in a negative image.

**Segment likelihood.** An image is represented by a set of segments $\{s\}$, and the set of all possible pairs of segments $\{c\}$. The image likelihood $p(I|\mathcal{M}; \mathbf{F}, a)$ remains as defined in equation (3), but

now $a = (a_1, a_2)$ specifies two foreground appearances, one for each segment in the pair. The likelihood of a segment $s$ is now defined as the maximum over all pairs containing it

$$p(s|\mathcal{M}; f, a) = \begin{cases} \max_{\{c|s \in c\}} p(c|\mathcal{M}, t) & \text{if } f = 1 \\ \beta & \text{if } f = 0 \end{cases} \qquad (5)$$

**Pair likelihood.** The observed variables in our model are segments $s$ and pairs of segments $c$. A pair $c = (s_1, s_2, \{c_r^k\})$ is defined by two segments $s_1, s_2$ and their relative geometric measurements $\{c_r^k\}$ (relative orientation and relative area in our implementation). The likelihood of a pair given the model is

$$p(c|\mathcal{M}, a) = \underbrace{p(s_{1,a}, s_{2,a}|a)}_{appearance} \cdot \underbrace{\prod_j \left( p(s_{1,g}^j|\Phi_1^j)^{v_1^j} \cdot p(s_{2,g}^j|\Phi_2^j)^{v_2^j} \right)}_{shape} \cdot \underbrace{\prod_k \left( p(c_r^k|\Psi^k)^{v_r^k} \right)}_{layout} \cdot p(c|\delta) \qquad (6)$$

The binary model parameters $\mathcal{M} = (\alpha, \beta, \delta, \{\lambda_1^j\}, \{\lambda_2^j\}, \{\gamma^k\})$ control the behavior of the pair likelihood. The two sets of $\lambda_i^j = (\Phi_i^j, v_i^j)$ are analogous to their counterparts in the unary model, and define the geometric distributions and their associated activation states for each segment in the pair respectively. The layout part of the model captures the interaction between the two segments in the pair. For each relative geometric property $\gamma^k = (\Psi^k, v_r^k)$ the model gives its distribution $\Psi^k$ over pairs of foreground segments and its activation state $v_r^k$. The model parameter $\delta$ determines whether the pattern is composed of pairs of *adjacent* segments ($\delta = 1$) or just any pair of segments ($\delta = 0$). The factor $p(c|\delta)$ is defined as 0 iff $\delta = 1$ and the segments in $c$ are not adjacent, while it is 1 in all other cases (so, when $\delta = 1$, $p(c|\delta)$ acts as a pair selector). The appearance factor $p(s_{1,a}, s_{2,a}|a) = [s_{1,a} = a_1 \wedge s_{2,a} = a_2]$ is 1 when the two segments have the foreground appearances $a = (a_1, a_2)$ for this image.

As an example, the model for a general stripe pattern is as follows. $\alpha = (\mathcal{A}, \mathcal{A})$ contains all pairs of appearances from $\mathcal{A}$. The geometric properties $\lambda_1^{elong}, \lambda_1^{curv}$ are active ($v_1^j = 1$) and their distributions $\Phi_1^j$ peaked at high elongation and low curvedness. The corresponding properties $\{\lambda_2^j\}$ have similar values. The layout parameters are $\delta = 1$, and $\gamma^{rel\_area}, \gamma^{rel\_orient}$ are active and peaked at 0 (expressing that the two segments are parallel and have the same area). Finally, $\beta$ is a value very close to 0, as the probability of a random segment under this complex model is very low.

## 4 Learning the model

**Image Likelihood.** The image likelihood defined in (3) depends on the foreground/background labels $\mathbf{F}$ and on the foreground appearance $a$. Computing the complete likelihood, given only the model $\mathcal{M}$, involves maximizing $a$ over the appearances $\alpha$ allowed by the model, and over $\mathbf{F}$:

$$p(I|\mathcal{M}) = \max_{a \in \alpha} \max_{\mathbf{F}} p(I|\mathcal{M}; \mathbf{F}, a) \qquad (7)$$

The maximization over $\mathbf{F}$ is easily achieved by setting each $f$ to the greater of the two cases in equation (4) (equation (5) for a binary model). The maximization over $a$ requires trying out all allowed appearances $\alpha$. This is computationally inexpensive, as typically there are about 32 entries in the appearance codebook.

**Training data.** We learn the model parameters in a weakly supervised setting. The training data consists of positive $\mathcal{I}_+ = \{I_+^i\}$ and negative images $\mathcal{I}_- = \{I_-^i\}$. While many of the positive images contain examples of the attribute to be learnt (figure 4), a considerable proportion don't. Conversely, some of the negative images do contain the attribute. Hence, we must operate under a weak assumption: the attribute occurs more frequently on positive training images than on negative. Moreover, only the (unreliable) image label is given, not the location of the attribute in the image. As demonstrated in section 5, our approach is able to learn from this noisy training data.

Although our attribute models are generative, learning them in a discriminative fashion greatly helps given the challenges posed by the weakly supervised setting. For example, in figure 4 most of the overall surface for images labeled 'red' is actually *white*. Hence, a maximum likelihood estimator over the positive training set alone would learn white, not red. A discriminative approach instead

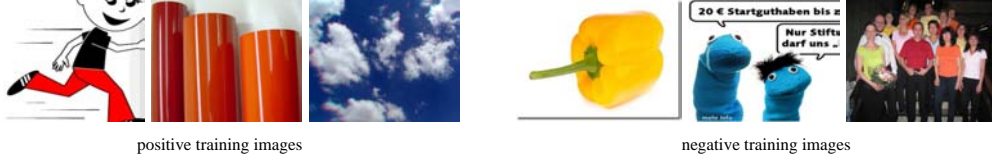

|  positive training images | negative training images |

Figure 4: *Advantages of discriminative training. The task is to learn the attribute 'red'. Although the most frequent color in the positive training images is white, white is also common across the negative set.*

notices that white occurs frequently also on the negative set, and hence correctly picks up red, as it is most discriminative for the positive set. Formally, the task of learning is to determine the model parameters $\mathcal{M}$ that maximize the likelihood ratio

$$\frac{p(\mathcal{I}_+|\mathcal{M})}{p(\mathcal{I}_-|\mathcal{M})} = \frac{\prod_{I_+^i \in \mathcal{I}_+} p(I_+^i|\mathcal{M})}{\prod_{I_-^i \in \mathcal{I}_-} p(I_-^i|\mathcal{M})} \tag{8}$$

**Learning procedure.** The parameters of the binary model are $\mathcal{M} = (\alpha, \beta, \delta, \{\lambda_1^j\}, \{\lambda_2^j\}, \{\gamma^k\})$, as defined in the previous sections. Since the binary model is a superset of the unary one, we only explain here how to learn the binary case. The procedure for the unary model is derived analogously. In our implementation, $\alpha$ can contain either a single appearance, or *all* appearances in the codebook $\mathcal{A}$. The former case covers attributes such as colors, or patterns with specific colors (such as zebra stripes). The latter case covers generic patterns, as it allows each image to pick a different appearance $a \in \alpha$, while at the same time it properly constrains all segments/pairs within an image to share the same appearance (e.g. subsequent pairs of stripe segments have the same appearance, forming a pattern of alternation ABABAB). Because of this definition, $\alpha$ can take on $(1 + |\mathcal{A}|)^2/2$ different values (sets of appearances). As typically a codebook of $|\mathcal{A}| \leq 32$ appearances is sufficient to model the data, we can afford exhaustive search over all possible values of $\alpha$. The same goes for $\delta$, which can only take on two values.

Given a fixed $\alpha$ and $\delta$, the learning task reduces to estimating the background probability $\beta$, and the geometric properties $\{\lambda_1^j\}, \{\lambda_2^j\}, \{\gamma^k\}$. To achieve this, we need determine the latent variable $\mathbf{F}$ for each training image, as it is necessary for estimating the geometric distributions over the foreground segments. These are in turn necessary for estimating $\beta$. Given $\beta$ and the geometric properties we can estimate $\mathbf{F}$ (equation (6)). This particular circular dependence in the structure of our model suggests a relatively simple and computationally cheap approximate optimization algorithm:

1. For each $I \in \{\mathcal{I}_+ \bigcup \mathcal{I}_-\}$, estimate an initial $\mathbf{F}$ and $a$ via equation (7), using an initial $\beta = 0.01$, and no geometry (i.e. all activation variables set to 0).

2. Estimate all geometric distributions $\Phi_1^j, \Phi_2^j, \Psi^k$ over the foreground segments/pairs from all images, according to the initial estimates $\{\mathbf{F}\}$.

3. Estimate $\beta$ and the geometric activations $v$ iteratively:

   (a) Update $\beta$ as the average probability of segments from $\mathcal{I}_-$. This is obtained using the foreground expression of (5) for *all* segments of $\mathcal{I}_-$.

   (b) Activate the geometric property which most increases the likelihood-ratio (8) (i.e. set the corresponding $v$ to 1). Stop iterating when no property increases (8).

4. The above steps already yield a reasonable estimate of all model parameters. We use it as initialization for the following EM-like iteration, which refines $\beta$ and $\Phi_1^j, \Phi_2^j, \Psi^k$

   (a) Update $\{\mathbf{F}\}$ given the current $\beta$ and geometric properties (set each $f$ to maximize (5))

   (b) Update $\Phi_1^j, \Phi_2^j, \Psi^k$ given the current $\{\mathbf{F}\}$.

   (c) Update $\beta$ over $\mathcal{I}_-$ using the current $\Phi_1^j, \Phi_2^j, \Psi^k$.

The algorithm is repeated over all possible $\alpha$ and $\delta$, and the model maximizing (8) is selected. Notice how $\beta$ is continuously re-estimated as more geometric properties are added. This implicitly offers to the selector the probability of an average negative segment under the current model as an up-to-date baseline for comparison. It prevents the model from overspecializing as it pushes it to only pick up properties which distinguish positive segments/pairs from negative ones.

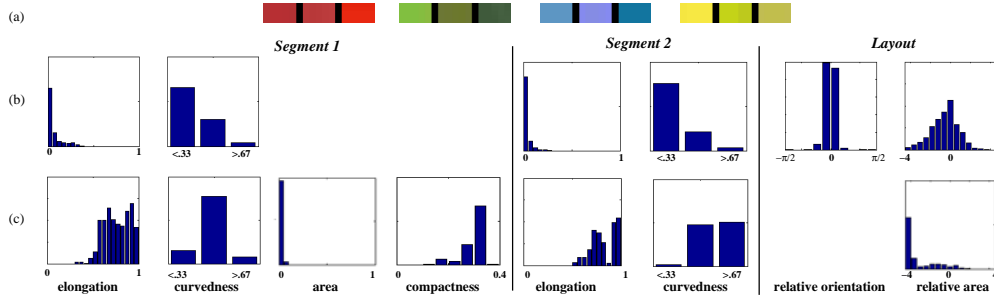

Figure 5: *a) color models learnt for red, green, blue, and yellow. For each, the three most frequent patch types are displayed. Notice how each model covers different shades of a color. b+c) geometric properties of the learned models for stripes (b) and dots (c). Both models are binary, have general appearance, i.e. $\alpha = (\mathcal{A}, \mathcal{A})$, and adjacent segments, i.e. $\delta = 1$. The figure shows the geometric distributions for the activated geometric properties. Lower elongation values indicate more elongated segments. A blank slot means the property is not active for that attribute. See main text for discussion.*

One last, implicit, parameter is the model complexity: is the attribute unary or binary ? This is tackled through model selection: we learn the best unary and binary models independently, and then select the one with highest likelihood-ratio. The comparison is meaningful because image likelihood is measured in the same way in both unary and binary cases (i.e. as the product over the segment probabilities, equation (3)).

## 5    Experimental results

**Learning.**    We present results on learning four colors (red, green, blue, and yellow) and three patterns (stripes, dots, and checkerboard). The positive training set for a color consists of the 14 images in the first page returned by Google-images when queried by the color name. The proportion of positive images unrelated to the color varies between $21\%$ and $36\%$, depending on the color (e.g. figure 4). The negative training set for a color contains all positive images for the other colors. Our approach delivers an excellent performance. In all cases, the correct model is returned: unary, no active geometric property, and the correct color as a specific appearance (figure 5a).

Stripes are learnt from 74 images collected from Google-images using 'striped', 'stripe', 'stripes' as queries. $20\%$ of them don't contain stripes. The positive training set for dots contains 35 images, $29\%$ of them without dots, collected from textile vendors websites and Google-images (keywords 'dots', 'dot', 'polka dots'). For both attributes, the $56$ images for colors act as negative training set. As shown in figure 5, the learnt models capture well the nature of these attributes. Both stripes and dots are learnt as binary and with general appearance, while they differ substantially in their geometric properties. Stripes are learnt as elongated, rather straight pairs of segments, with largely the same properties for the two segments in a pair. Their layout is meaningful as well: adjacent, nearly parallel, and with similar area. In contrast, dots are learnt as small, unelongated, rather curved segments, embedded within a much larger segment. This can be seen in the distribution of the area of the first segment, the dot, relative to the area of the second segment, the 'background' on which dots lie. The background segments have a very curved, zigzagging outline, because they circumvent several dots. In contrast to stripes, the two segments that form this dotted pattern are not symmetric in their properties. This characterisic is modeled well by our approach, confirming its flexibility. We also train a model from the first 22 Google-images for the query 'checkerboard', $68\%$ of which show a black/white checkerboard. The learnt model is binary, with one segment for a black square and the other for an adjacent white square, demonstrating the learning algorithm correctly infers both models with specific and generic appearance, adapting to the training data.

**Recognition.**    Once a model is learnt, it can be used to recognize whether a novel image contains the attribute, by computing the likelihood (7). Moreover, the area covered by the attribute is localized by the segments with $f = 1$ (figure 6). We report results for red, yellow, stripes, and dots. All test images are downloaded from Yahoo-images, Google-images, and Flickr. There are 45 (red), 39 (yellow), 106 (stripes), 50 (dots) positive test images. In general, the object carrying the attribute stands against a background, and often there are other objects in the image, making the localization task non-trivial. Moreover, the images exhibit extreme variability: there are paintings as well as photographs, stripes appear in any orientation, scale, and appearance, and they are often are deformed

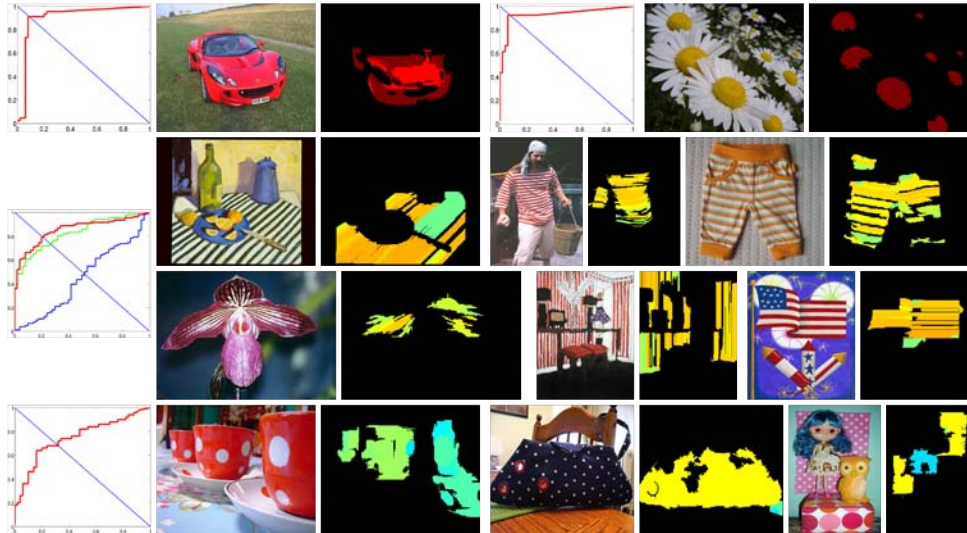

Figure 6: *Recognition results. Top row: red (left) and yellow (right). Middle rows: stripes. Bottom row: dots. We give a few example test images and the corresponding localizations produced by the learned models. Segments are colored according to their foreground likelihood, using matlab's jet colormap (from dark blue to green to yellow to red to dark red). Segments deemed not to belong to the attribute are not shown (black). In the case of dots, notice how the pattern is formed by the dots themselves* and *by the uniform area on which they lie. The ROC plots shows the image classification performance for each attribute. The two lower curves in the stripes plot correspond to a model without layout, and without either layout nor any geometry respectively. Both curves are substantially lower, confirming the usefulness of the layout and shape components of the model.*

(human body poses, animals, etc.). The same goes for dots, which can vary in thickness, spacing, and so on. Each positive set is coupled with a negative one, in which the attribute doesn't appear, composed of 50 images from the Caltech-101 'Things' set [12]. Because these negative images are rich in colors, textures and structure, they pose a considerable challenge for the classification task.

As can be seen in figure 6, our method achieves accurate localizations of the region covered by the attribute. The behavior on stripe patterns composed of more than two appearances is particularly interesting (the trousers in the rightmost example). The model explains them as disjoint groups of binary stripes, with the two appearances which cover the largest image area. In terms of recognizing whether an image contains the attribute, the method performs very well for red and yellow, with ROC equal-error rates above $90\%$. Performance is convincing also for stripes and dots, especially since these attributes have generic appearance, and hence must be recognized based only on geometry and layout. In contrast, colors enjoy a very distinctive, specific appearance.

## Footnotes

*This research was supported by the EU project CLASS. The authors thank Dr. Josef Sivic for fruitful discussions and helpful comments on this paper.

## References

[1] N. Dalal and B. Triggs, *Histograms of Oriented Gradients for Human Detection*, CVPR, 2005.

[2] P. Felzenszwalb and D Huttenlocher, *Efficient Graph-Based Image Segmentation*, IJCV, (50):2, 2004.

[3] R. Fergus, P. Perona, and A. Zisserman, Object Class Recognition by Unsupervised Scale-Invariant Learning, CVPR, 2003.

[4] N. Jojic and Y. Caspi, *Capturing image structure with probabilistic index maps*, CVPR, 2004

[5] S. Lazebnik, C. Schmid, and J. Ponce, *A Sparse Texture Representation Using Local Affine Regions*, PAMI, (27):8, 2005

[6] Y. Liu, Y. Tsin, and W. Lin, *The Promise and Perils of Near-Regular Texture*, IJCV, (62):1, 2005

[7] J. Van de Weijer, C. Schmid, and J. Verbeek, *Learning Color Names from Real-World Images*, CVPR, 2007.

[8] M. Varma and A. Zisserman, *Texture classification: Are filter banks necessary?*, CVPR, 2003.

[9] J. Winn, A. Criminisi, and T. Minka, *Object Categorization by Learned Universal Visual Dictionary*, ICCV, 2005.

[10] J. Winn and N. Jojic. *LOCUS: Learning Object Classes with Unsupervised Segmentation*, ICCV, 2005.

[11] K. Yanai and K. Barnard, *Image Region Entropy: A Measure of "Visualness" of Web Images Associated with One Concept*, ACM Multimedia, 2005.

[12] Caltech 101 dataset: www.vision.caltech.edu/Image_Datasets/Caltech101/Caltech101.html
